# Proximal Newton-type methods for convex optimization

**Jason D. Lee**[*] **and Yuekai Sun**[*]
Institute for Computational and Mathematical Engineering
Stanford University, Stanford, CA
{jdl17,yuekai}@stanford.edu

**Michael A. Saunders**
Department of Management Science and Engineering
Stanford University, Stanford, CA
saunders@stanford.edu

## Abstract

We seek to solve convex optimization problems in *composite form*:

$$\underset{x \in \mathbb{R}^n}{\text{minimize}}\ f(x) := g(x) + h(x),$$

where $g$ is convex and continuously differentiable and $h : \mathbb{R}^n \to \mathbb{R}$ is a convex but not necessarily differentiable function whose proximal mapping can be evaluated efficiently. We derive a generalization of Newton-type methods to handle such convex but nonsmooth objective functions. We prove such methods are globally convergent and achieve superlinear rates of convergence in the vicinity of an optimal solution. We also demonstrate the performance of these methods using problems of relevance in machine learning and statistics.

## 1 Introduction

Many problems of relevance in machine learning, signal processing, and high dimensional statistics can be posed in *composite form*:

$$\underset{x \in \mathbb{R}^n}{\text{minimize}}\ f(x) := g(x) + h(x), \tag{1}$$

where $g : \mathbb{R}^n \to \mathbb{R}$ is a convex, continuously differentiable loss function, and $h : \mathbb{R}^n \to \mathbb{R}$ is a convex, continuous, but not necessarily differentiable penalty function. Such problems include: (i) the *lasso* [23] (ii) multitask learning [14] and (iii) trace-norm matrix completion [6].

We describe a family of Newton-type methods tailored to these problems that achieve superlinear rates of convergence subject to standard assumptions. These methods can be interpreted as generalizations of the classic proximal gradient method that use the curvature of the objective function to select a search direction.

### 1.1 First-order methods

The most popular methods for solving convex optimization problems in composite form are first-order methods that use proximal mappings to handle the nonsmooth part. SpaRSA is a generalized *spectral projected gradient* method that uses a *spectral step length* together with a *nonmonotone line*

---

[*]Equal contributors

*search* to improve convergence [24]. TRIP by Kim et al. also uses a spectral step length but selects search directions using a trust-region strategy [12]. TRIP performs comparably with SpaRSA and the projected Newton-type methods we describe later.

A closely related family of methods is the set of *optimal first-order methods*, also called *accelerated first-order methods*, which achieve $\epsilon$-suboptimality within $O(1/\sqrt{\epsilon})$ iterations [22]. The two most popular methods in this family are Auslender and Teboulle's method [1] and Fast Iterative Shrinkage-Thresholding Algorithm (FISTA), by Beck and Teboulle [2]. These methods have been implemented in the software package TFOCS and used to solve problems that commonly arise in statistics, machine learning, and signal processing [3].

## 1.2 Newton-type methods

There are three classes of methods that generalize Newton-type methods to handle nonsmooth objective functions. The first are projected Newton-type methods for constrained optimization [20]. Such methods cannot handle nonsmooth objective functions; they tackle problems in composite form via constraints of the form $h(x) \leq \tau$. PQN is an implementation that uses a limited-memory quasi-Newton update and has both excellent empirical performance and theoretical properties [19, 18].

The second class of these methods by Yu et al. [25] uses a local quadratic approximation to the smooth part of the form

$$Q(x) := f(x) + \sup_{g \in \partial f(x)} g^T d + \frac{1}{2} d^T H d,$$

where $\partial f(x)$ denotes the subdifferential of $f$ at $x$. These methods achieve state-of-the-art performance on many problems of relevance, such as $\ell_1$-regularized logistic regression and $\ell_2$-regularized support vector machines.

This paper focuses on proximal Newton-type methods that were previously studied in [16, 18] and are closely related to the methods of Fukushima and Mine [10] and Tseng and Yun [21]. Both use search directions $\Delta x$ that are solutions to subproblems of the form

$$\underset{d}{\text{minimize}} \ \nabla g(x)^T d + \frac{1}{2} d^T H d + h(x + d),$$

where $H$ is a positive definite matrix that approximates the Hessian $\nabla^2 g(x)$. Fukushima and Mine choose $H$ to be a multiple of the identity, while Tseng and Yun set some components of the search direction $\Delta x$ to be zero to obtain a (block) coordinate descent direction. Proximal Newton-type methods were first studied empirically by Mark Schmidt in his Ph.D. thesis [18].

The methods GLMNET [9] ($\ell_1$-regularized regression), LIBLINEAR [26] ($\ell_1$-regularized classification), QUIC and recent work by Olsen et al. [11, 15] (sparse inverse covariance estimation) are special cases of proximal Newton-type methods. These methods are considered state-of-the-art for their specific applications, often outperforming generic methods by orders of magnitude. QUIC and LIBLINEAR also achieve a quadratic rate of convergence, although these results rely crucially on the structure of the $\ell_1$ norm and do not generalize to generic nonsmooth regularizers.

The quasi-Newton splitting method developed by Becker and Fadili is equivalent to a proximal quasi-Newton method with rank-one Hessian approxiamtion [4]. In this case, they can solve the subproblem via the solution of a single variable root finding problem, making their method significantly more efficient than a generic proximal Newton-type method.

The methods described in this paper are a special case of *cost approximation* (CA), a class of methods developed by Patriksson [16]. CA requires a CA function $\varphi$ and selects search directions via subproblems of the form

$$\underset{d}{\text{minimize}} \ g(x) + \varphi(x + d) - \varphi(x) + h(x + d) - \nabla g(x)^T d.$$

Cost approximation attains a linear convergence rate. Our methods are equivalent to using the CA function $\varphi(x) := \frac{1}{2} x^T H x$. We refer to [16] for details about cost approximation and its convergence analysis.

## 2 Proximal Newton-type methods

We seek to solve convex optimization problems in composite form:

$$\underset{x \in \mathbb{R}^n}{\text{minimize}} \ f(x) := g(x) + h(x). \tag{2}$$

We assume $g : \mathbb{R}^n \to \mathbb{R}$ is a closed, proper convex, continuously differentiable function, and its gradient $\nabla g$ is Lipschitz continuous with constant $L_1$; *i.e.*

$$\|\nabla g(x) - \nabla g(y)\| \leq L_1 \|x - y\|$$

for all $x$ and $y$ in $\mathbb{R}^n$. $h : \mathbb{R}^n \to \mathbb{R}$ is a closed and proper convex but not necessarily everywhere differentiable function whose *proximal mapping* can be evaluated efficiently. We also assume the optimal value, $f^\star$, is attained at some optimal solution $x^\star$, not necessarily unique.

### 2.1 The proximal gradient method

The *proximal mapping* of a convex function $h$ at $x$ is

$$\text{prox}_h(x) = \arg\min_y \ h(y) + \frac{1}{2}\|y - x\|^2.$$

Proximal mappings can be interpreted as generalized projections because if $h$ is the indicator function of a convex set, then $\text{prox}_h(x)$ is the projection of $x$ onto the set.

The classic proximal gradient method for composite optimization uses proximal mappings to handle the nonsmooth part of the objective function and can be interpreted as minimizing the nonsmooth function $h$ plus a simple quadratic approximation to the smooth function $g$ during every iteration:

$$x_{k+1} = \text{prox}_{t_k h}\left(x_k - t_k \nabla g(x_k)\right)$$

$$= \arg\min_y \ \nabla g(x_k)^T (y - x_k) + \frac{1}{2 t_k}\|y - x_k\|^2 + h(y),$$

where $t_k$ denotes the $k$-th step length. We can also interpret the proximal gradient step as a *generalized gradient step*

$$G_f(x) = \text{prox}_h(x - \nabla g(x)) - x. \tag{3}$$

$G_f(x) = 0$ if and only if $x$ minimizes $f$ so $\|G_f(x)\|$ generalizes the smooth first-order measure of optimality $\|\nabla f(x)\|$.

Many state-of-the-art methods for problems in composite form, such as SpaRSA and the optimal first-order methods, are variants of this method. Our method uses a Newton-type approximation in lieu of the simple quadratic to achieve faster convergence.

### 2.2 The proximal Newton iteration

**Definition 2.1** (Scaled proximal mappings). *Let $h$ be a convex function and $H$, a positive definite matrix. Then the scaled proximal mapping of $h$ at $x$ is defined to be*

$$\text{prox}_h^H(x) := \arg\min_y \ h(y) + \frac{1}{2}\|y - x\|_H^2. \tag{4}$$

Proximal Newton-type methods use the iteration

$$x_{k+1} = x_k + t_k \Delta x_k, \tag{5}$$

$$\Delta x_k := \text{prox}_h^{H_k}\left(x_k - H_k^{-1}\nabla g(x_k)\right) - x_k, \tag{6}$$

where $t_k > 0$ is the $k$-th step length, usually determined using a line search procedure and $H_k$ is an approximation to the Hessian of $g$ at $x_k$. We can interpret the search direction $\Delta x_k$ as a step to the minimizer of the nonsmooth function $h$ plus a local quadratic approximation to $g$ because

$$\text{prox}_h^{H_k}\left(x_k - H_k^{-1}\nabla g(x_k)\right)$$

$$= \arg\min_y \ h(y) + \frac{1}{2}\|(y - x_k) + H_k^{-1}\nabla g(x_k)\|_{H_k}^2$$

$$= \arg\min_y \ \nabla g(x_k)^T (y - x_k) + \frac{1}{2}(y - x_k)^T H_k(y - x_k) + h(y). \tag{7}$$

Hence, the search direction solves the subproblem

$$\Delta x_k = \arg\min_d \ \nabla g(x_k)^T d + \frac{1}{2} d^T H_k d + h(x_k + d)$$
$$= \arg\min_d \ Q_k(d) + h(x_k + d).$$

To simplify notation, we shall drop the subscripts and say $x^+ = x + t\Delta x$ in lieu of $x_{k+1} = x_k + t_k \Delta x_k$ when discussing a single iteration.

**Lemma 2.2** (Search direction properties). *If $H$ is a positive definite matrix, then the search direction $\Delta x = \arg\min_d Q(d) + h(x + d)$ satisfies:*

$$f(x^+) \leq f(x) + t\left(\nabla g(x)^T \Delta x + h(x + \Delta x) - h(x)\right) + O(t^2), \tag{8}$$

$$\nabla g(x)^T \Delta x + h(x + \Delta x) - h(x) \leq -\Delta x^T H \Delta x. \tag{9}$$

Lemma 2.2 implies the search direction is a descent direction for $f$ because we can substitute (9) into (8) to obtain

$$f(x^+) \leq f(x) - t\Delta x^T H \Delta x + O(t^2).$$

We use a quasi-Newton approximation to the Hessian and a first-order method to solve the subproblem for a search direction, although the user is free to use a method of his or her choice. Empirically, we find that inexact solutions to the subproblem yield viable descent directions.

We use a backtracking line search to select a step length $t$ that satisfies a sufficient descent condition:

$$f(x^+) \leq f(x) + \alpha t \Delta \tag{10}$$

$$\Delta := \nabla g(x)^T \Delta x + h(x + \Delta x) - h(x), \tag{11}$$

where $\alpha \in (0, 0.5)$. This sufficient descent condition is motivated by our convergence analysis but it also seems to perform well in practice.

**Lemma 2.3** (Step length conditions). *Suppose $H \succeq mI$ for some $m > 0$ and $\nabla g$ is the Lipschitz continuous with constant $L_1$. Then the step lengths*

$$t \leq \min\left\{1, \frac{2m}{L_1}(1 - \alpha)\right\}. \tag{12}$$

*satisfies the sufficient descent condition* (10).

---

**Algorithm 1** A generic proximal Newton-type method

---

**Require:** $x_0$ in dom $f$
 1: **repeat**
 2:     Update $H_k$ using a quasi-Newton update rule
 3:     $z_k \leftarrow \text{prox}_h^{H_k}\left(x_k - H_k^{-1}\nabla g(x_k)\right)$
 4:     $\Delta x_k \leftarrow z_k - x_k$
 5:     Conduct backtracking line search to select $t_k$
 6:     $x_{k+1} \leftarrow x_k + t_k \Delta x_k$
 7: **until** stopping conditions are satisfied

---

## 3  Convergence analysis

### 3.1  Global convergence

We assume our Hessian approximations are sufficiently positive definite; *i.e.* $H_k \succeq mI$, $k = 1, 2, \ldots$ for some $m > 0$. This assumption guarantees the existence of step lengths that satisfy the sufficient decrease condition.

**Lemma 3.1** (First-order optimality conditions). *Suppose $H$ is a positive definite matrix. Then $x$ is a minimizer of $f$ if and only if the search direction is zero at $x$; i.e.*

$$0 = \arg\min_d \ Q(d) + h(x + d).$$

The global convergence of proximal Newton-type methods results from the fact that the search directions are descent directions and if our Hessian approximations are sufficiently positive definite, then the step lengths are bounded away from zero.

**Theorem 3.2** (Global convergence). *Suppose $H_k \succeq mI$, $k = 1, 2, \ldots$ for some $m > 0$. Then the sequence $\{x_k\}$ generated by a proximal Newton-type method converges to a minimizer of $f$.*

## 3.2 Convergence rate

If $g$ is twice-continuously differentiable and we use the second order Taylor approximation as our local quadratic approximation to $g$, then we can prove $\{x_k\}$ converges Q-quadratically to the optimal solution $x^\star$. We assume in a neighborhood of $x^\star$: (i) $g$ is strongly convex with constant $m$; *i.e.*

$$\nabla^2 g(x) \succeq mI, \ x \in N_\epsilon(x^\star)$$

where $N_\epsilon(x^\star) := \{x \mid \|x - x^\star\| \le \epsilon\}$; and (ii) $\nabla^2 g$ is Lipschitz continuous with constant $L_2$.

This convergence analysis is similar to that of Fukushima and Miné [10] and Patriksson [16]. First, we state two lemmas: (i) that says step lengths of unity satisfy the sufficient descent condition after sufficiently many iterations and (ii) that the backward step is nonexpansive.

**Lemma 3.3.** *Suppose (i) $\nabla^2 g \succeq mI$ and (ii) $\nabla^2 g$ is Lipschitz continuous with constant $L_2$. If we let $H_k = \nabla^2 g(x_k)$, $k = 1, 2, \ldots$, then the step length $t_k = 1$ satisfies the sufficient decrease condition (10) for $k$ sufficiently large.*

We can characterize the solution of the subproblem using the first-order optimality conditions for (4). Let $y$ denote $\text{prox}_h^H\left(x - H^{-1}\nabla g(x)\right)$, then

$$H(x - H^{-1}\nabla g(x) - y) \in \partial h(u).$$

or equivalently

$$[H - \nabla g](x) \in [H + \partial h](y)$$

Let $R(x)$ and $S(x)$ denote $\left[\frac{1}{m}(H + \partial h)\right]^{-1}(x)$ and $\left[\frac{1}{m}(H - \nabla g)\right](x)$ respectively, where $m$ is the smallest eigenvalue of $H$. Then

$$y = [H + \partial h]^{-1}[H - \nabla g](x) = R \circ S(x).$$

.

**Lemma 3.4.** *Suppose $R(x) = \left[\frac{1}{m}H + \partial h\right]^{-1}(x)$, where $H$ is positive definite. Then $R$ is firmly-nonexpansive; i.e. for $x$ and $y$ in $\text{dom } f$, $R$ satisfies*

$$(R(x) - R(y))^T(x - y) \ge \|R(x) - R(y)\|^2.$$

We note that $x^\star$ is a fixed point of $R \circ S$; *i.e.* $R \circ S(x^\star) = x^\star$, so we can express $\|y - x^\star\|$ as

$$\|y - x^\star\| = \|R \circ S(x) - R \circ S(x^\star)\| \le \|S(x) - S(x^\star)\|.$$

**Theorem 3.5.** *Suppose (i) $\nabla^2 g \succeq mI$ and (ii) $\nabla^2 g$ is Lipschitz continuous with constant $L_2$. If we let $H_k = \nabla^2 g(x_k)$, $k = 1, 2, \ldots$, then $\{x_k\}$ converges to $x^\star$ Q-quadratically; i.e.*

$$\frac{\|x_{k+1} - x^\star\|}{\|x_k - x^\star\|^2} \to c.$$

We can also use the fact that the proximal Newton method converges quadratically to prove a proximal quasi-Newton method converges superlinearly. We assume the quasi-Newton Hessian approximations satisfy the Dennis-Moré criterion [7]:

$$\frac{\left\|\left(H_k - \nabla^2 g(x^\star)\right)(x_{k+1} - x_k)\right\|}{\|x_{k+1} - x_k\|} \to 0. \tag{13}$$

We first prove two lemmas: (i) step lengths of unity satisfy the sufficient descent condition after sufficiently many iterations and (ii) the proximal quasi-Newton step is close to the proximal Newton step.

**Lemma 3.6.** *Suppose $g$ is twice-continuously differentiable and the eigenvalues of $H_k$, $k = 1, 2, \ldots$ are bounded; i.e. there exist $M \geq m > 0$ such that $mI \preceq H \preceq MI$. If $\{H_k\}$ satisfy the Dennis-Moré criterion, then the unit step length satisfies the sufficient descent condition (10) after sufficiently many iterations.*

**Lemma 3.7.** *Suppose $H$ and $\hat{H}$ are positive definite matrices with bounded eigenvalues; i.e. $mI \preceq H \preceq MI$ and $\hat{m}I \preceq \hat{H} \preceq \hat{M}I$. Let $\Delta x$ and $\Delta \hat{x}$ denote the search directions generated using $H$ and $\hat{H}$ respectively; i.e.*

$$\Delta x = \text{prox}_h^H \left( x - H^{-1} \nabla g(x) \right) - x,$$
$$\Delta \hat{x} = \text{prox}_h^{\hat{H}} \left( x - \hat{H}^{-1} \nabla g(x) \right) - x.$$

*Then these two search directions satisfy*

$$\|\Delta x - \Delta \hat{x}\| \leq \sqrt{\frac{1 + c(H, \hat{H})}{m}} \left\| (\hat{H} - H) \Delta x \right\|^{1/2} \|\Delta x\|^{1/2},$$

*where $c$ is a constant that depends on $H$ and $\hat{H}$.*

**Theorem 3.8.** *Suppose $g$ is twice-continuously differentiable and the eigenvalues of $H_k$, $k = 1, 2, \ldots$ are bounded. If $\{H_k\}$ satisfy the Dennis-Moré criterion, then the sequence $\{x_k\}$ converges to $x^\star$ Q-superlinearly; i.e.*

$$\frac{\|x_{k+1} - x^\star\|}{\|x_k - x^\star\|} \to 0.$$

# 4 Computational experiments

## 4.1 PNOPT: Proximal Newton OPTimizer

PNOPT[1] is a MATLAB package that uses proximal Newton-type methods to minimize convex objective functions in composite form. PNOPT can build BFGS and L-BFGS approximation to the Hessian (the user can also supply a Hessian approximation) and uses our implementation of SpaRSA or an optimal first order method to solve the subproblem for a search direction.

PNOPT uses an early stopping condition for the subproblem solver based on two ideas: (i) the subproblem should be solved to a higher accuracy if $Q_k$ is a good approximation to $g$ and (ii) near a solution, the subproblem should be solved almost exactly to achieve fast convergence.

We thus require that the solution to the $k$-th subproblem (7) $y_k^\star$ satisfy

$$\|G_{Q+h}(y_k^\star)\| \leq \eta_k \|G_f(y_k^\star)\|, \tag{14}$$

where $G_f(x)$ denotes the generalized gradient step at $x$ (3) and $\eta_k$ is a *forcing term*. We choose forcing terms based on the agreement between $g$ and the previous quadratic approximation to $g$ $Q_{k-1}$. We set $\eta_1 := 0.5$ and

$$\eta_k := \min \left\{ 0.5, \frac{\|\nabla g(x_k) - \nabla Q_{k-1}(x_k)\|}{\|\nabla g(x_k)\|} \right\}, \ k = 2, 3, \ldots \tag{15}$$

This choice measures the agreement between $\nabla g(x_k)$ and $\nabla Q_{k-1}(x_k)$ and is borrowed from a choice of forcing terms for inexact Newton methods described by Eisenstat and Walker [8]. Empirically, we find that this choice avoids "oversolving" the subproblem and yields desirable convergence behavior.

We compare the performance of PNOPT, our implementation of SpaRSA, and the TFOCS implementations of Auslender and Teboulle's method (AT) and FISTA on $\ell_1$-regularized logistic regression and Markov random field structure learning. We used the following settings:

1. PNOPT: We use an L-BFGS approximation to the Hessian with $L = 50$ and set the sufficient decrease parameter to $\alpha = 0.0001$. To solve the subproblem, we use the TFOCS implementation of FISTA.

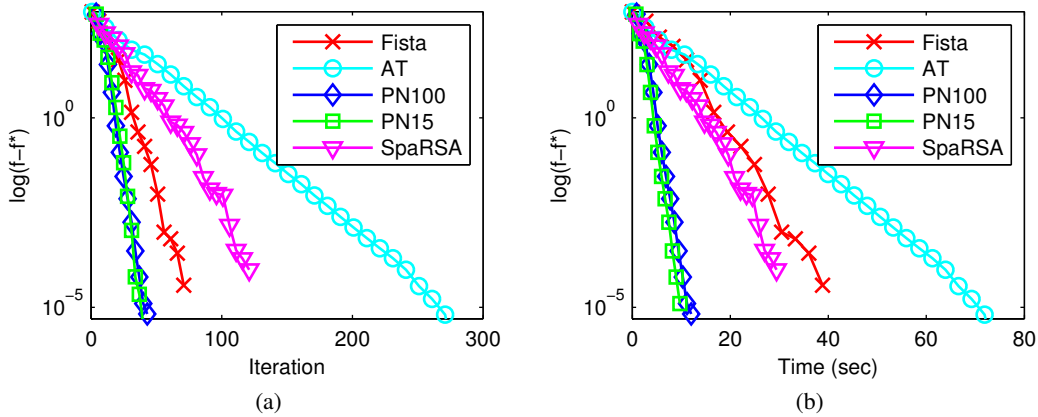

Figure 1: Figure 1a and 1b compare two variants of proximal Newton-type methods with SpaRSA and TFOCS on on the MRF structure learning problem.

2. SpaRSA: We use a nonmonotone line search with a 10 iteration memory and also set the sufficient decrease parameter to $\alpha = 0.0001$. Our implementation of SpaRSA is included in PNOPT as the default solver for the subproblem.

3. AT/FISTA: We set `tfocsOpts.restart = -inf` to turn on adaptive restarting and use default values for the rest of the settings.

These experiments were conducted on a machine running the 64-bit version of Ubuntu 12.04 with an Intel Core i7 870 CPU and 8 GB RAM.

## 4.2 Markov random field structure learning

We seek the maximum likelihood estimates of the parameters of a Markov random field (MRF) subject to a group elastic-net penalty on the estimates. The objective function is given by

$$\underset{\theta}{\text{minimize}} \ -\sum_{(r,j)\in E} \theta_{rj}(x_r, x_j) + \log Z(\theta) + \sum_{(r,j)\in E} \left( \lambda_1 \left\| \theta_{rj} \right\|_2 + \lambda_2 \left\| \theta_{rj} \right\|_F^2 \right). \quad (16)$$

$x_r$ is a $k$ state variable; $x_j$ is a $l$ state variable, and each parameter block $\theta_{rj}$ is a $k \times l$ matrix that is associated with an edge in the MRF. We randomly generate a graphical model with $|V| = 12$ and $n = 300$. The edges are sample uniformly with $p = 0.3$. The parameters of the non-zero edges are sampled from a $\mathcal{N}(0, 1)$ distribution.

The group elastic-net penalty regularizes the solution and promotes solutions with a few non-zero groups $\theta_{rj}$ corresponding to edges of the graphical model [27]. The regularization parameters were set to $\lambda_1 = \sqrt{n \log |V|}$ and $\lambda_2 = .1\lambda_1$. These parameter settings are shown to be model selection consistent under certain irrepresentable conditions [17].

The algorithms for solving (16) require evaluating the value and gradient of the smooth part. For a discrete graphical model without special structure, the smooth part requires $O(k^{|V|})$ operations to evaluate, where $k$ is the number of states per variable. Thus even for our small example, where $k = 3$ and $|V| = 12$, function and gradient evaluations dominate the computational expense required to solve (16).

We see that for maximum likelihood learning in graphical models, it is important to minimize the number of function evaluations. Proximal Newton-type methods are well-suited to solve such problems because the main computational expense is shifted to solving the subproblems that do not require function evaluations.

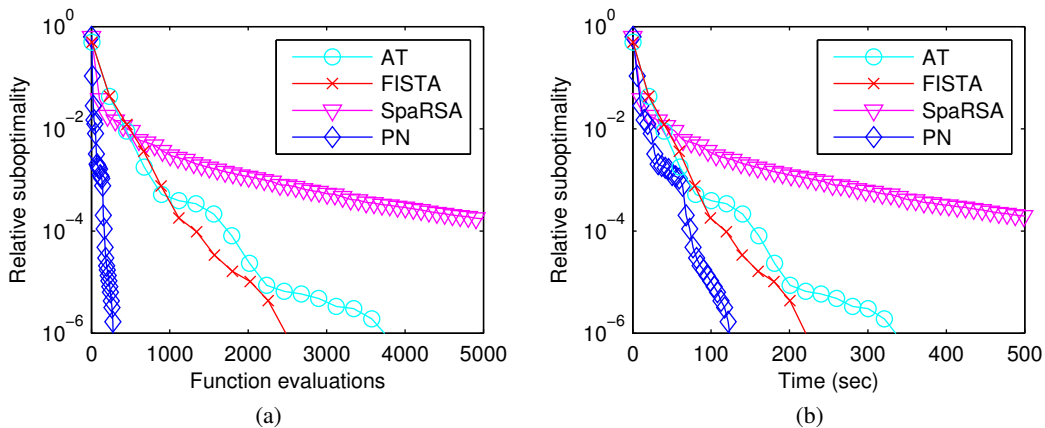

Figure 2: Figure 2 compares proximal Newton-type methods with SpaRSA and TFOCS on $\ell_1$-regularized logistic regression.

### 4.3 $\ell_1$-regularized logistic regression

Given training data $(x_i, y_i)$, $i = 1, 2, \ldots, n$, $\ell_1$-regularized logistic regression trains a classifier via the solution of the convex optimization problem

$$\underset{w \in \mathbb{R}^p}{\text{minimize}} \quad \frac{1}{n} \sum_{i=1}^{n} \log(1 + \exp(-y_i w^T x_i)) + \lambda \|w\|_1. \tag{17}$$

for a set of parameters $w$ in $\mathbb{R}^p$. The regularization term $\|w\|_1$ avoids overfitting the training data and promotes sparse solutions. $\lambda$ is trades-off between goodness-of-fit and model complexity.

We use the dataset `gisette`, a handwritten digits dataset from the NIPS 2003 feature selection challenge. The dataset is available at `http://www.csie.ntu.edu.tw/~cjlin/libsvmtools/datasets`. We train our classifier using the original training set consisting of 6000 examples starting at $w = 0$. $\lambda$ was chosen to match the value reported in [26], where it was chosen by five-fold cross validation on the training set.

The `gisette` dataset is quite dense (3 million nonzeros in the $6000 \times 5000$ design matrix) and the evaluation of the log-likelihood requires many expensive exp/log operations. We see in figure 2 that PNOPT outperforms the other methods because the computational expense is shifted to solving the subproblems, whose objective functions are cheap to evaluate.

## 5 Conclusion

Proximal Newton-type methods are natural generalizations of first-order methods that account for curvature of the objective function. They share many of the desirable characteristics of traditional first-order methods for convex optimization problems in composite form and achieve superlinear rates of convergence subject to standard assumptions. These methods are especially suited to problems with expensive function evaluations because the main computational expense is shifted to solving subproblems that do not require function evaluations.

## 6 Acknowledgements

We wish to thank Trevor Hastie, Nick Henderson, Ernest Ryu, Ed Schmerling, Carlos Sing-Long, and Walter Murray for their insightful comments.

## Footnotes

[1]PNOPT is available at www.stanford.edu/group/SOL/software/pnopt.html.

# References

[1] A. Auslender and M. Teboulle, *Interior gradient and proximal methods for convex and conic optimization*, SIAM J. Optim., 16 (2006), pp. 697–725.

[2] A. Beck and M. Teboulle , *A fast iterative shrinkage-thresholding algorithm for linear inverse problems*, SIAM J. Imaging Sci., 2 (2009), pp. 183–202.

[3] S. R. Becker, M. J. Candès, and M. C. Grant, *Templates for convex cone problems with applications to sparse signal recovery*, Math. Program. Comput., 3 (2011), pp. 1–54.

[4] S. Becker and J. Fadili, *A quasi-Newton proximal splitting method*, NIPS, Lake Tahoe, California, 2012.

[5] S. Boyd and L. Vandenberghe, *Convex Optimization*, Cambridge University Press, Cambridge, 2004.

[6] E. J. Candès and B. Recht, *Exact matrix completion via convex optimization*, Found. Comput. Math, 9 (2009), pp. 717–772.

[7] J. E. Dennis, Jr. and J. J. Moré, *A characterization of superlinear convergence and its application to quasi-Newton methods*, Math. Comp., 28, (1974), pp. 549–560.

[8] S. C. Eisenstat and H. F. Walker, *Choosing the forcing terms in an inexact Newton method*, SIAM J. Sci. Comput., 17 (1996), pp. 16–32.

[9] J. Friedman, T. Hastie, H. Holfing, and R. Tibshirani, *Pathwise coordinate optimization*, Ann. Appl. Stat. (2007), pp. 302–332

[10] M. Fukushima and H. Mine, *A generalized proximal point algorithm for certain non-convex minimization problems*, Internat. J. Systems Sci., 12 (1981), pp. 989–1000.

[11] C. J. Hsieh, M. A. Sustik, P. Ravikumar, and I. S. Dhillon, *Sparse inverse covariance matrix estimation using quadratic approximation*, NIPS, Granada, Spain, 2011.

[12] D. Kim, S. Sra, and I. S. Dhillon, *A scalable trust-region algorithm with applications to mixed-norm regression*, ICML, Haifa, Israel, 2010.

[13] Y. Nesterov, *Gradient methods for minimizing composite objective function*, CORE discussion paper, 2007.

[14] G. Obozinski, B. Taskar, and M. I. Jordan, *Joint covariate selection and joint subspace selection for multiple classification problems*, Stat. Comput. (2010), pp. 231–252

[15] P. Olsen, F. Oztoprak, J. Nocedal, S. Rennie, *Newton-like methods for sparse inverse covariance estimation*, NIPS, Lake Tahoe, California, 2012.

[16] M. Patriksson, *Nonlinear Programming and Variational Inequality Problems*, Kluwer Academic Publishers, The Netherlands, 1999.

[17] P. Ravikumar, M. J. Wainwright and J. D. Lafferty, *High-dimensional Ising model selection using $\ell 1$-regularized logistic regression*, Ann. Statist. (2010), pp. 1287-1319.

[18] M. Schmidt, *Graphical Model Structure Learning with l1-Regularization*, Ph.D. Thesis (2010), University of British Columbia

[19] M. Schmidt, E. van den Berg, M. P. Friedlander, and K. Murphy, *Optimizing costly functions with simple constraints: a limited-memory projected quasi-Newton algorithm*, AISTATS, Clearwater Beach, Florida, 2009.

[20] M. Schmidt, D. Kim, and S. Sra, *Projected Newton-type methods in machine learning*, in  S. Sra, S. Nowozin, and S. Wright, editors, Optimization for Machine Learning, MIT Press (2011).

[21] P. Tseng and S. Yun, *A coordinate gradient descent method for nonsmooth separable minimization*, Math. Prog. Ser. B, 117 (2009), pp. 387–423.

[22] P. Tseng, *On accelerated proximal gradient methods for convex-concave optimization*, submitted to SIAM J. Optim. (2008).

[23] R. Tibshirani, *Regression shrinkage and selection via the lasso*, J. R. Stat. Soc. Ser. B Stat. Methodol., 58 (1996), pp. 267–288.

[24] S. J. Wright, R. D. Nowak, and M. A. T. Figueiredo, *Sparse reconstruction by separable approximation*, IEEE Trans. Signal Process., 57 (2009), pp. 2479–2493.

[25] J. Yu, S. V. N. Vishwanathan, S. Günter, and N. N. Schraudolph, *A Quasi-Newton Approach to Nonsmooth Convex Optimization*, ICML, Helsinki, Finland, 2008.

[26] G. X. Yuan, C. H. Ho and C. J. Lin, *An improved GLMNET for $\ell 1$-regularized logistic regression and support vector machines*, National Taiwan University, Tech. Report 2011.

[27] R. H. Zou and T. Hastie, *Regularization and variable selection via the elastic net*, J. R. Stat. Soc. Ser. B Stat. Methodol., 67 (2005), pp. 301–320.

